# A Generalized Natural Actor-Critic Algorithm

**Tetsuro Morimura**[†]**, Eiji Uchibe**[‡]**, Junichiro Yoshimoto**[‡]**, Kenji Doya**[‡]

†: IBM Research – Tokyo, Kanagawa, Japan
‡: Okinawa Institute of Science and Technology, Okinawa, Japan
tetsuro@jp.ibm.com, {uchibe,jun-y,doya}@oist.jp

## Abstract

Policy gradient Reinforcement Learning (RL) algorithms have received substantial attention, seeking stochastic policies that maximize the average (or discounted cumulative) reward. In addition, extensions based on the concept of the Natural Gradient (NG) show promising learning efficiency because these regard metrics for the task. Though there are two candidate metrics, Kakade's Fisher Information Matrix (FIM) for the policy (action) distribution and Morimura's FIM for the state-action joint distribution, but all RL algorithms with NG have followed Kakade's approach. In this paper, we describe a generalized Natural Gradient (gNG) that linearly interpolates the two FIMs and propose an efficient implementation for the gNG learning based on a theory of the estimating function, the *generalized Natural Actor-Critic* (gNAC) algorithm. The gNAC algorithm involves a near optimal auxiliary function to reduce the variance of the gNG estimates. Interestingly, the gNAC can be regarded as a natural extension of the current state-of-the-art NAC algorithm [1], as long as the interpolating parameter is appropriately selected. Numerical experiments showed that the proposed gNAC algorithm can estimate gNG efficiently and outperformed the NAC algorithm.

## 1 Introduction

Policy Gradient Reinforcement Learning (PGRL) attempts to find a policy that maximizes the average (or time-discounted) reward, based on gradient ascent in the policy parameter space [2, 3, 4]. Since it is possible to handle the parameters controlling the randomness of the policy, the PGRL, rather than the value-based RL, can find the appropriate stochastic policy and has succeeded in several practical applications [5, 6, 7]. However, depending on the tasks, PGRL methods often require an excessively large number of learning time-steps to construct a good stochastic policy, due to the learning plateau where the optimization process falls into a stagnant state, as was observed for a very simple Markov Decision Process (MDP) with only two states [8]. In this paper, we propose a new PGRL algorithm, a *generalized Natural Actor-Critic* (gNAC) algorithm, based on the natural gradient [9].

Because "natural gradient" learning is the steepest gradient method in a Riemannian space and the direction of the natural gradient is defined on that metric, it is an important issue how to design the Riemannian metric. In the framework of PGRL, the stochastic policies are represented as parametric probability distributions. Thus the Fisher Information Matrices (FIMs) with respect to the policy parameter induce appropriate Riemannian metrics. Kakade [8] used an average FIM for the policy over the states and proposed a *natural policy gradient* (NPG) learning. Kakade's FIM has been widely adopted and various algorithms for the NPG learning have been developed by many researchers [1, 10, 11]. These are based on the actor-critic framework, called the natural actor-critic (NAC) [1]. Recently, the concept of "*Natural State-action Gradient*" (NSG) learning has been proposed in [12], which shows potential to reduce the learning time spent by being better at avoiding the learning plateaus than the NPG. This natural gradient is on the FIM of the state-action joint distribution as the Riemannian metric for RL, which is directly associated with the average rewards as the objective function. Morimura et al. [12] showed that the metric of the NSG corresponds with

the changes in the stationary state-action joint distribution. In contrast, the metric of the NPG takes into account only changes in the action distribution and ignores changes in the state distribution, which also depends on the policy in general. They also showed experimental results with exact gradients where the NSG learning outperformed NPG learning, especially with large numbers of states in the MDP. However, no algorithm for estimating the NSG has been proposed, probably because the estimation for the derivative of log stationary state distribution was difficult [13]. Therefore, the development of a tractable algorithm for NSG would be of great importance, and this is the one of the primary goals of this paper.

Meanwhile, it would be very difficult to select an appropriate FIM because it would be dependent on the given task. Accordingly, we created a linear interpolation of both of the FIMs as a generalized Natural Gradient (gNG) and derived an efficient approach to estimate the gNG by applying the theory of the estimating function for stochastic models [14] in Section 3. In Section 4, we derive a gNAC algorithm with an instrumental variable, where a policy parameter is updated by a gNG estimate that is a solution of the estimating function derived in Section 3, and show that the gNAC can be regarded as a natural extension of the current state-of-the-art NAC algorithm [1]. To validate the performance of the proposed algorithm, numerical experiments are shown in Section 5, where the proposed algorithm can estimate the gNG efficiently and outperformed the NAC algorithm [1].

## 2 Background of Policy Gradient and Natural Gradient for RL

We briefly review the policy gradient and natural gradient learning as gradient ascent methods for RL and also present the motivation of the gNAC approach.

### 2.1 Policy Gradient Reinforcement Learning

PGRL is modeled on a discrete-time Markov Decision Process (MDP) [15, 16]. It is defined by the quintuplet $(\mathcal{S}, \mathcal{A}, p, r, \pi)$, where $\mathcal{S} \ni s$ and $\mathcal{A} \ni a$ are finite sets of states and actions, respectively. Also, $p : \mathcal{S} \times \mathcal{A} \times \mathcal{S} \rightarrow [0, 1]$ is a state transition probability function of a state $s$, an action $a$, and the following state $s_{+1}$, i.e.[1], $p(s_{+1}|s, a) \triangleq \Pr(s_{+1}|s, a)$. $R : \mathcal{S} \times \mathcal{A} \times \mathcal{S} \rightarrow \mathcal{R}$ is a bounded reward function of $s$, $a$, and $s_{+1}$, which defines an immediate reward $r = R(s, a, s_{+1})$ observed by a learning agent at each time step. The action probability function $\pi : \mathcal{A} \times \mathcal{S} \times \mathcal{R}^d \rightarrow [0, 1]$ uses $a$, $s$, and a policy parameter $\boldsymbol{\theta} \in \mathcal{R}^d$ to define the decision-making rule of the learning agent, which is also called a policy, i.e., $\pi(a|s; \boldsymbol{\theta}) \triangleq \Pr(a|s, \boldsymbol{\theta})$. The policy is normally parameterized by users and is controlled by tuning $\boldsymbol{\theta}$. Here, we make two assumptions in the MDP.

**Assumption 1** *The policy is always differentiable with respect to $\boldsymbol{\theta}$ and is non-redundant for the task, i.e., the statistics $\overline{\boldsymbol{F}}_a(\boldsymbol{\theta}) \in \mathcal{R}^{d \times d}$ (defined in Section 2.2) are always bounded and non-singular.*

**Assumption 2** *The Markov chain $\mathrm{M}(\boldsymbol{\theta}) \triangleq \{\mathcal{S}, \mathcal{A}, p, \pi, \boldsymbol{\theta}\}$ is always ergodic (irreducible and aperiodic).*

Under Assumption 2, there exists a unique stationary state distribution $d_{\boldsymbol{\theta}}(s) \triangleq \Pr(s|\mathrm{M}(\boldsymbol{\theta}))$, which is equal to the limiting distribution and independent of the initial state, $d_{\boldsymbol{\theta}}(s') = \lim_{t \to \infty} \Pr(S_{+t} = s'|S = s, \mathrm{M}(\boldsymbol{\theta}))$, $\forall s \in \mathcal{S}$. This distribution satisfies the balance equation: $d_{\boldsymbol{\theta}}(s_{+1}) = \sum_{s \in \mathcal{S}} \sum_{a \in \mathcal{A}} p(s_{+1}|s, a)\pi(a|s; \boldsymbol{\theta})d_{\boldsymbol{\theta}}(s)$.

The goal of PGRL is to find the policy parameter $\boldsymbol{\theta}^*$ that maximizes the average of the immediate rewards, the *average reward*,

$$\eta(\boldsymbol{\theta}) \triangleq \mathbb{E}_{\boldsymbol{\theta}}[r] = \sum_{s \in \mathcal{S}} \sum_{a \in \mathcal{A}} \sum_{s_{+1} \in \mathcal{S}} d_{\boldsymbol{\theta}}(s)\pi(a|s; \boldsymbol{\theta})p(s_{+1}|s, a)R(s, a, s_{+1}), \tag{1}$$

where $\mathbb{E}_{\boldsymbol{\theta}}[a]$ denotes the expectation of $a$ on the Markov chain $\mathrm{M}(\boldsymbol{\theta})$. The derivative of the average reward for (1) with respect to the policy parameter, $\nabla_{\boldsymbol{\theta}}\eta(\boldsymbol{\theta}) \triangleq [\partial\eta(\boldsymbol{\theta})/\partial\theta_1, ..., \partial\eta(\boldsymbol{\theta})/\partial\theta_d]^\top$, which is referred to as the Policy Gradient (PG), is

$$\nabla_{\boldsymbol{\theta}}\eta(\boldsymbol{\theta}) = \mathbb{E}_{\boldsymbol{\theta}}\left[ r\nabla_{\boldsymbol{\theta}}\ln\{d_{\boldsymbol{\theta}}(s)\pi(a|s; \boldsymbol{\theta})\} \right].$$

Therefore, the average reward $\eta(\boldsymbol{\theta})$ will be increased by updating the policy parameter as $\boldsymbol{\theta} := \boldsymbol{\theta} + \alpha \nabla_{\boldsymbol{\theta}} \eta(\boldsymbol{\theta})$, where $:=$ denotes the right-to-left substitution and $\alpha$ is a sufficiently small learning rate. This framework is called the PGRL [4].

It is noted that the ordinary PGRL methods omit the differences in sensitivities and the correlations between the elements of $\boldsymbol{\theta}$, as defined by the probability distributions of the MDP, while most probability distributions expressed in the MDP have some form of a manifold structure instead of a Euclidean structure. Accordingly, the updating direction of the policy parameter by the ordinary gradient method will be different from the steepest directions on these manifolds. Therefore, the optimization process sometimes falls into a stagnant state, commonly called a *plateau* [8, 12].

## 2.2 Natural Gradients for PGRL

To avoid the plateau problem, the concept of the natural gradient was proposed by Amari [9], which is a gradient method on a Riemannian space. The parameter space being a Riemannian space implies that the parameter $\boldsymbol{\theta} \in \mathcal{R}^d$ is on the manifold with the Riemannian metric $\boldsymbol{G}(\boldsymbol{\theta}) \in \mathcal{R}^{d \times d}$ (a semi-positive definite matrix), instead of being on a Euclidean manifold of an arbitrarily parameterized policy, and the squared length of a small incremental vector $\Delta\boldsymbol{\theta}$ connecting $\boldsymbol{\theta}$ to $\boldsymbol{\theta} + \Delta\boldsymbol{\theta}$ is given by $\|\Delta\boldsymbol{\theta}\|_{\boldsymbol{G}}^2 = \Delta\boldsymbol{\theta}^{\top}\boldsymbol{G}(\boldsymbol{\theta})\Delta\boldsymbol{\theta}$, where $\top$ denotes the transpose. Under the constraint $\|\Delta\boldsymbol{\theta}\|_{\boldsymbol{G}}^2 = \varepsilon^2$ for a sufficiently small constant $\varepsilon$, the steepest ascent direction of the function $\eta(\boldsymbol{\theta})$ on the manifold $\boldsymbol{G}(\boldsymbol{\theta})$ is given by

$$\widetilde{\nabla}_{\boldsymbol{G}(\boldsymbol{\theta})} \eta(\boldsymbol{\theta}) = \boldsymbol{G}(\boldsymbol{\theta})^{-1} \nabla_{\boldsymbol{\theta}} \eta(\boldsymbol{\theta}),$$

which is called the natural gradient (NG). Accordingly, to (locally) maximize $\eta(\boldsymbol{\theta})$, $\boldsymbol{\theta}$ is incrementally updated with

$$\boldsymbol{\theta} := \boldsymbol{\theta} + \alpha \, \widetilde{\nabla}_{\boldsymbol{G}(\boldsymbol{\theta})} \eta(\boldsymbol{\theta}).$$

The direction of the NG is defined using a Riemannian metric. Thus, an appropriate choice of the Riemannian metric for the task is required. With RL, two kinds of Fisher Information Matrices (FIMs) $\boldsymbol{F}(\boldsymbol{\theta})$ have been proposed as the Riemannian metric matrices $\boldsymbol{G}(\boldsymbol{\theta})$:[2]

(I) Kakade [8] focuses only on the changes in the policy (action) distributions and proposes defining the metric matrix with the notation $\nabla_{\boldsymbol{\theta}} a_{\boldsymbol{\theta}} b_{\boldsymbol{\theta}} \triangleq (\nabla_{\boldsymbol{\theta}} a_{\boldsymbol{\theta}}) b_{\boldsymbol{\theta}}$, as

$$\overline{\boldsymbol{F}}_a(\boldsymbol{\theta}) \triangleq \mathbb{E}_{\boldsymbol{\theta}} \left[ \nabla_{\boldsymbol{\theta}} \ln\pi(a|s;\boldsymbol{\theta}) \nabla_{\boldsymbol{\theta}} \ln\pi(a|s;\boldsymbol{\theta})^{\top} \right] = \mathbb{E}_{\boldsymbol{\theta}} \left[ \boldsymbol{F}_a(\boldsymbol{\theta}, s) \right], \tag{2}$$

where $\boldsymbol{F}_a(\boldsymbol{\theta}, s) \triangleq \mathbb{E}_{\boldsymbol{\theta}}[\nabla_{\boldsymbol{\theta}} \ln\pi(a|s;\boldsymbol{\theta}) \nabla_{\boldsymbol{\theta}} \ln\pi(a|s;\boldsymbol{\theta})^{\top}|s]$ is the FIM of the policy at a state $s$. The NG on this FIM, $\widetilde{\nabla}_{\overline{\boldsymbol{F}}_a(\boldsymbol{\theta})} \eta(\boldsymbol{\theta}) = \overline{\boldsymbol{F}}_a(\boldsymbol{\theta})^{-1} \nabla_{\boldsymbol{\theta}} \eta(\boldsymbol{\theta})$, is called the Natural Policy Gradient (NPG).

(II) Considering that the average reward $\eta(\boldsymbol{\theta})$ in (1) is affected not only by the policy distributions $\pi(a|s;\boldsymbol{\theta})$ but also by the stationary state distribution $d_{\boldsymbol{\theta}}(s)$, Moimura et al. [12] proposed the use of the FIM of the state-action joint distribution for RL,

$$\boldsymbol{F}_{s,a}(\boldsymbol{\theta}) \triangleq \mathbb{E}_{\boldsymbol{\theta}} \left[ \nabla_{\boldsymbol{\theta}} \ln \left\{ d_{\boldsymbol{\theta}}(s)\pi(a|s;\boldsymbol{\theta}) \right\} \nabla_{\boldsymbol{\theta}} \ln \left\{ d_{\boldsymbol{\theta}}(s)\pi(a|s;\boldsymbol{\theta}) \right\}^{\top} \right] = \boldsymbol{F}_s(\boldsymbol{\theta}) + \overline{\boldsymbol{F}}_a(\boldsymbol{\theta}), \tag{3}$$

where $\boldsymbol{F}_s(\boldsymbol{\theta}) \triangleq \sum_{s \in \mathcal{S}} d_{\boldsymbol{\theta}}(s) \nabla_{\boldsymbol{\theta}} \ln d_{\boldsymbol{\theta}}(s) \nabla_{\boldsymbol{\theta}} \ln d_{\boldsymbol{\theta}}(s)^{\top}$ is the FIM of $d_{\boldsymbol{\theta}}(s)$. The NG on this FIM, $\widetilde{\nabla}_{\boldsymbol{F}_{s,a}(\boldsymbol{\theta})} \eta(\boldsymbol{\theta}) = \boldsymbol{F}_{s,a}(\boldsymbol{\theta})^{-1} \nabla_{\boldsymbol{\theta}} \eta(\boldsymbol{\theta})$, is called the Natural State-action Gradient (NSG).

Some algorithms for the NPG learning, such as NAC [1] and NTD [10, 11], can be successfully implemented using modifications of the actor-critic frameworks based on the LSTD$Q(\lambda)$ [18] and TD$(\lambda)$ [16]. In contrast, no tractable algorithm for the NSG learning has been proposed to date. However, it has been suggested that the NSG learning it better than the NPG learning due to the three differences [12]: (a) The NSG learning appropriately benefits from the concepts of Amari's NG learning, since the metric $\boldsymbol{F}_{s,a}(\boldsymbol{\theta})$ necessarily and sufficiently accounts for the probability distribution that the average reward depends on. (b) $\boldsymbol{F}_{s,a}(\boldsymbol{\theta})$ is an analogy to the Hessian matrix of the average reward. (c) Numerical experiments show a strong tendency to avoid entrapment in a learning plateau[3], especially with large numbers of states. Therefore, the development of a tractable algorithm for NSG is important, and this is one of the goals of our work.

On the other hand, it was proven that the metric of NPG learning, $\overline{\boldsymbol{F}}_a(\boldsymbol{\theta})$, accounts for the infinite time-steps joint distribution in the Markov chain $\mathrm{M}(\boldsymbol{\theta})$ [19, 1], while the metric of NSG learning, $\boldsymbol{F}_{s,a}(\boldsymbol{\theta})$ accounts only for the single time-step distribution, which is the stationary state-action joint distribution $d_{\boldsymbol{\theta}}(s)\pi(a|s;\boldsymbol{\theta})$. Accordingly, the mixing time of $\mathrm{M}(\boldsymbol{\theta})$ might be drastically changed with NSG learning compared to NPG learning, since the mixing time depends on the multiple (not necessarily infinite) time-steps rather than the single time-step, i.e., while various policies can lead to the same stationary state distribution, Markov chains associated with these policies have different mixing times. A larger mixing time makes it difficult for the learning agent to explore the environment and to estimate the gradient with finite samples. The ranking of the performances of the NPG and NSG learning will be dependent on the RL task properties. Thus, we consider a mixture of NPG and NSG as a generalized NG (gNG) and propose the approach of '*generalized Natural Actor-Critic*' (gNAC), in which the policy parameter of an actor is updated by an estimate of the gNG of a critic.

## 3 Generalized Natural Gradient for RL

First we explain the definition and properties of the generalized Natural Gradient (gNG). Then we introduce the estimating functions to build up a foundation for an efficient estimation of the gNG.

### 3.1 Definition of gNG for RL

In order to define an interpolation between NPG and NSG with a parameter $\iota \in [0, 1]$, we consider a linear interpolation from the FIM of (2) for the NPG to the FIM of (3) for the NSG, written as

$$\tilde{\boldsymbol{F}}_{s,a}(\boldsymbol{\theta}, \iota) \triangleq \iota \boldsymbol{F}_s(\theta) + \overline{\boldsymbol{F}}_a(\boldsymbol{\theta}). \tag{4}$$

Then the natural gradient of the interpolated FIM is

$$\widetilde{\nabla}_{\tilde{\boldsymbol{F}}_{s,a}(\boldsymbol{\theta},\iota)} \eta(\boldsymbol{\theta}) = \tilde{\boldsymbol{F}}_{s,a}(\boldsymbol{\theta}, \iota)^{-1} \nabla_{\boldsymbol{\theta}} \eta(\boldsymbol{\theta}), \tag{5}$$

which we call the *"generalized natural gradient"* for RL with the interpolating parameter $\iota$, gNG($\iota$). Obviously, gNG($\iota\!=\!0$) and gNG($\iota\!=\!1$) are equivalent to the NPG and the NSG, respectively. When $\iota$ is equal to $1/t$, this FIM $\tilde{\boldsymbol{F}}_{s,a}(\boldsymbol{\theta}, \iota)$ is equivalent to the FIM of the $t$ time-steps joint distribution from the stationary state distribution $d_{\boldsymbol{\theta}}(s)$ on $\mathrm{M}(\boldsymbol{\theta})$ [12]. Thus, this interpolation controlled by $\iota$ can be interpreted as a continuous interpolation with respect to the time-steps of the joint distribution, so that $\iota : 1 \to 0$ is inversely proportional to $t : 1 \to \infty$. The term '*generalized*' of gNG($\iota$) reflects the generalization as the time steps on the joint distribution that the NG follows.

### 3.2 Estimating Function of gNG($\iota$)

We provide a general view of the estimation of the gNG($\iota$) using the theory of the estimating function, which provides well-established results for parameter estimation [14].

Such a function $\boldsymbol{g} \in \mathcal{R}^d$ for an estimator $\boldsymbol{\omega} \in \mathcal{R}^d$ (and a variable $x$) is called an estimating function when it satisfies these conditions for all $\boldsymbol{\theta}$:

$$\mathbb{E}_{\boldsymbol{\theta}} \left[ \boldsymbol{g}(x, \boldsymbol{\omega}^*) \right] = \mathbf{0} \tag{6}$$

$$\det \left| \mathbb{E}_{\boldsymbol{\theta}} [\nabla_{\boldsymbol{\omega}} \boldsymbol{g}(x, \boldsymbol{\omega})] \right| \neq 0, \quad {}^{\forall}\boldsymbol{\omega} \tag{7}$$

$$\mathbb{E}_{\boldsymbol{\theta}} \left[ \boldsymbol{g}(x, \boldsymbol{\omega})^{\top} \boldsymbol{g}(x, \boldsymbol{\omega}) \right] < \infty, \quad {}^{\forall}\boldsymbol{\omega} \tag{8}$$

where $\boldsymbol{\omega}^*$ and $\det | \cdot |$ denote the exact solution of this estimation and the determinant, respectively.

**Proposition 1** *The $d$-dimensional (random) function*

$$\boldsymbol{g}'_{\iota,\boldsymbol{\theta}}(s, a; \boldsymbol{\omega}) \triangleq \nabla_{\boldsymbol{\theta}}\ln\{d_{\boldsymbol{\theta}}(s)\pi(a|s;\boldsymbol{\theta})\} \left( r - \nabla_{\boldsymbol{\theta}}\ln\{d_{\boldsymbol{\theta}}(s)^{\iota}\pi(a|s;\boldsymbol{\theta})\}^{\top} \boldsymbol{\omega} \right) \tag{9}$$

*is an estimating function for gNG($\iota$), such that the unique solution of $\mathbb{E}_{\boldsymbol{\theta}}[\boldsymbol{g}'_{\iota,\boldsymbol{\theta}}(s, a; \boldsymbol{\omega})] = \mathbf{0}$ with respect to $\boldsymbol{\omega}$ is equal to the gNG($\iota$).*

**Proof:** From (1) and (4), the equation

$$\mathbf{0} = \mathbb{E}_{\boldsymbol{\theta}}[\boldsymbol{g}'_{\iota,\boldsymbol{\theta}}(s, a; \boldsymbol{\omega}^*)] = \nabla_{\boldsymbol{\theta}}\eta(\boldsymbol{\theta}) - \tilde{\boldsymbol{F}}_{s,a}(\boldsymbol{\theta}, \iota)\boldsymbol{\omega}^*$$

holds. Thus, $\boldsymbol{\omega}^*$ is equal to the gNG($\iota$) from (5). The remaining conditions from (7) and (8), which the estimating function must satisfy, also obviously hold (under Assumption 1). $\qquad\square$

In order to estimate gNG($\iota$) by using the estimating function (9) with finite $T$ samples on M($\boldsymbol{\theta}$), the simultaneous equation

$$\frac{1}{T}\sum_{t=0}^{T-1} \boldsymbol{g}'_{\iota,\boldsymbol{\theta}}(s_t, a_t; \widehat{\boldsymbol{\omega}}) = \boldsymbol{0}$$

is solved with respect to $\boldsymbol{\omega}$. The solution $\widehat{\boldsymbol{\omega}}$, which is also called the M-estimator [20], is an unbiased estimate of gNG($\iota$), so that $\widehat{\boldsymbol{\omega}} = \boldsymbol{\omega}^*$ holds in the limit as $T \to \infty$.

Note that the conduct of solving the estimating function of (9) is equivalent to the linear regression with the instrumental variable $\nabla_{\boldsymbol{\theta}}\ln\{d_{\boldsymbol{\theta}}(s)\pi(a|s;\boldsymbol{\theta})\}$ where the regressand, the regressor, and the model parameter (estimator) are $r$ (or $\overline{R}(s,a)$), $\nabla_{\boldsymbol{\theta}}\ln\{d_{\boldsymbol{\theta}}(s)^{\iota}\pi(a|s;\boldsymbol{\theta})\}$, and $\boldsymbol{\omega}$, respectively [21], so that the regression residuals '$r - \nabla_{\boldsymbol{\theta}}\ln\{d_{\boldsymbol{\theta}}(s)^{\iota}\pi(a|s;\boldsymbol{\theta})\}^{\top}\boldsymbol{\omega}$' are not correlated with the instrumental variables $\nabla_{\boldsymbol{\theta}}\ln\{d_{\boldsymbol{\theta}}(s)\pi(a|s;\boldsymbol{\theta})\}$.

### 3.3   Auxiliary Function of Estimating Function

Although we made a simple algorithm implementing the gNAC approach with the M-estimator of the estimating function in (9), the performance of the estimation of gNG($\iota$) may be unacceptable for real RL applications, since the variance of the estimates of gNG($\iota$) tends to become too large. For that reason, we extend the estimating function using (9) by embedding an auxiliary function to create space for improvement in (9).

**Lemma 1** *The d-dimensional (random) function is an estimating function for gNG($\iota$),*

$$\boldsymbol{g}_{\iota,\boldsymbol{\theta}}(s,a;\boldsymbol{\omega}) \triangleq \nabla_{\boldsymbol{\theta}}\ln\{d_{\boldsymbol{\theta}}(s)\pi(a|s;\boldsymbol{\theta})\}\left(r - \nabla_{\boldsymbol{\theta}}\ln\{d_{\boldsymbol{\theta}}(s)^{\iota}\pi(a|s;\boldsymbol{\theta})\}^{\top}\boldsymbol{\omega} - \rho(s,s_{+1})\right), \qquad (10)$$

*where $\rho(s, s_{+1})$ is called the auxiliary function for (9):*

$$\rho(s, s_{+1}) \triangleq c + b(s) - b(s_{+1}). \qquad (11)$$

*The $c$ and $b(s)$ are an arbitrary bounded constant and an arbitrary bounded function of the state. respectively.*

**Proof:** See supplementary material.

Let $\mathcal{G}_{\boldsymbol{\theta}}$ denote the class of such functions $\boldsymbol{g}_{\boldsymbol{\theta}}$ with various auxiliary functions $\rho$. An optimal auxiliary function, which leads to minimizing the variance of the gNG estimate $\widehat{\boldsymbol{\omega}}$, is defined by the optimality criterion of the estimating functions [22]. An estimating function $\boldsymbol{g}^*_{\iota,\boldsymbol{\theta}}$ is optimal in $\mathcal{G}_{\iota,\boldsymbol{\theta}}$ if $\det|\boldsymbol{\Sigma}_{\boldsymbol{g}^*_{\iota,\boldsymbol{\theta}}}| \leq \det|\boldsymbol{\Sigma}_{\boldsymbol{g}_{\iota,\boldsymbol{\theta}}}|$ where $\boldsymbol{\Sigma}_{\boldsymbol{g}_{\boldsymbol{\theta}}} \triangleq \mathbb{E}_{\boldsymbol{\theta}}\left[\boldsymbol{g}_{\iota,\boldsymbol{\theta}}(s,a;\boldsymbol{\omega}^*)\boldsymbol{g}_{\iota,\boldsymbol{\theta}}(s,a;\boldsymbol{\omega}^*)^{\top}\right]$.

**Lemma 2** *Let us approximate (or assume)*

$$r \approx \mathbb{E}_{\boldsymbol{\theta}}[R(s,a,s_{+1})|s,a] \triangleq \overline{R}(s,a), \qquad (12)$$

$$\rho(s,s_{+1}) \approx \mathbb{E}_{\boldsymbol{\theta}}[\rho(s,s_{+1})|s,a] \quad \triangleq \overline{\rho}(s,a). \qquad (13)$$

*If the policy is non-degenerate for the task (so the dimension of $\boldsymbol{\theta}$, d, is equal to $\sum_{i=1}^{|\mathcal{S}|}(|\mathcal{A}_i| - 1)$, where $|\mathcal{S}|$ and $|\mathcal{A}_i|$ are the numbers of states and the available actions at state $s_i$, respectively) and $\boldsymbol{\omega}^*$ denotes the gNG($\iota$), then the 'near' optimal auxiliary function $\rho^*$ in the 'near' optimal estimating function $\boldsymbol{g}^*_{\iota,\boldsymbol{\theta}}(s,a;\boldsymbol{\omega})$ satisfies[4]*

$$\overline{R}(s,a) = \nabla_{\boldsymbol{\theta}}\ln\{d_{\boldsymbol{\theta}}(s)^{\iota}\pi(a|s;\boldsymbol{\theta})\}^{\top}\boldsymbol{\omega}^* + \mathbb{E}_{\boldsymbol{\theta}}[\rho^*(s,s_{+1})|s,a]. \qquad (14)$$

**Proof Sketch:** The covariance matrix for the criterion of the auxiliary function $\rho$ is approximated as

$$\boldsymbol{\Sigma}_{\boldsymbol{g}_{\boldsymbol{\theta}}} \approx \mathbb{E}_{\boldsymbol{\theta}}\left[\nabla_{\boldsymbol{\theta}}\ln\{d_{\boldsymbol{\theta}}(s)\pi(a|s;\boldsymbol{\theta})\}\nabla_{\boldsymbol{\theta}}\ln\{d_{\boldsymbol{\theta}}(s)\pi(a|s;\boldsymbol{\theta})\}^{\top}(\overline{R}(s,a) - \nabla_{\boldsymbol{\theta}}\ln\{d_{\boldsymbol{\theta}}(s)^{\iota}\pi(a|s;\boldsymbol{\theta})\}^{\top}\boldsymbol{\omega}^* - \overline{\rho}(s,a))^2\right]$$

$$\triangleq \hat{\boldsymbol{\Sigma}}_{\boldsymbol{g}_{\boldsymbol{\theta}}}. \qquad (15)$$

The function $\overline{\rho}(s,a)$ usually has $|\mathcal{S}|$ degrees of freedom over all of the $(s,a)$ couplets with the ergodicity of M($\boldsymbol{\theta}$), because "$b(s) - b(s_{+1})$" in $\rho$ has $(|\mathcal{S}| - 1)$ degrees of freedom over all of the $(s, s_{+1})$

couplets. The value of $\nabla_{\boldsymbol{\theta}}\ln\{d_{\boldsymbol{\theta}}(s)^{\iota}\pi(a|s;\boldsymbol{\theta})\}^{\top}\boldsymbol{\omega}$ has $\sum_{i=1}^{|\mathcal{S}|}(|\mathcal{A}_i|-1)$ degrees of freedom. $\overline{R}(s,a)$ has at most $\sum_{i=1}^{|\mathcal{S}|}|\mathcal{A}_i|$ degrees of freedom. Therefore, there exist $\rho^*$ and $\nabla_{\boldsymbol{\theta}}\ln\{d_{\boldsymbol{\theta}}(s)^{\iota}\pi(a|s;\boldsymbol{\theta})\}^{\top}\boldsymbol{\omega}^{\star}$ that satisfy (14). Remembering that $\nabla_{\boldsymbol{\theta}}\ln\{d_{\boldsymbol{\theta}}(s)^{\iota}\pi(a|s;\boldsymbol{\theta})\}^{\top}\boldsymbol{\omega}^*$ is the approximator of $\overline{R}(s,a)$ (or $r$) and $\boldsymbol{\omega}^*$ is independent of the choice of $\rho$ due to Lemma 1, we know that $\boldsymbol{\omega}^{\star}=\boldsymbol{\omega}^*$ holds. Therefore, if the estimating function has an auxiliary function $\rho^*$ satisfying (14), the criterion of the optimality for $\rho$ is minimized for $\det|\widehat{\boldsymbol{\Sigma}}_{\boldsymbol{g}_{\boldsymbol{\theta}}^*}|=0$ due to (15). $\qquad\square$

From Lemma 2, the near optimal auxiliary function $\rho^*$ can be regarded as minimizing the mean squared residuals to zero between $\overline{R}(s,a)$ and the estimator $\widehat{R}_{\rho}(s,a;\boldsymbol{\omega})\triangleq$ $\nabla_{\boldsymbol{\theta}}\ln\{d_{\boldsymbol{\theta}}(s)^{\iota}\pi(a|s;\boldsymbol{\theta})\}^{\top}\boldsymbol{\omega}+\rho(s,s_{+1})$. Thus, the meaning of this near optimality of $\boldsymbol{g}_{\iota,\boldsymbol{\theta}}^*(s,a;\widehat{\boldsymbol{\omega}})$ is interpreted as a near minimization of the Euclidean distance between $r$ and its approximator $\widehat{R}_{\rho*}(s,a;\widehat{\boldsymbol{\omega}})$, so that $\rho^*$ works to reduce the distance of the regressand $r$ and the subspace of the regressor $\nabla_{\boldsymbol{\theta}}\ln\{d_{\boldsymbol{\theta}}(s)^{\iota}\pi(a|s;\boldsymbol{\theta})\}$ of the M-estimator $\widehat{\boldsymbol{\omega}}$. In particular, $\overline{R}(s,a)$ is almost in this subspace at the point $\widehat{\boldsymbol{\omega}}=\boldsymbol{\omega}^*$. Lemma 2 leads directly to Corollary 1.

**Corollary 1** *Let $b_{\iota=0}^*(s)$ and $c_{\iota=0}^*$ be the functions in the near optimal auxiliary function $\rho^*(s,s_{+1})$ at $\iota=0$, then $b_{\iota=0}^*(s)$ and $c_{\iota=0}^*$ are equal to the (un-discounted) state value function [23] and the average reward, respectively.*

**Proof:** For all $s$, $\boldsymbol{\omega}$, and $\boldsymbol{\theta}$, the following equation holds,
$$\mathbb{E}_{\boldsymbol{\theta}}\left[\nabla_{\boldsymbol{\theta}}\ln\{d_{\boldsymbol{\theta}}(s)^{\iota=0}\pi(a|s;\boldsymbol{\theta})\}^{\top}\boldsymbol{\omega}\mid s\right]=\boldsymbol{\omega}^{\top}\sum_{a\in\mathcal{A}}\nabla_{\boldsymbol{\theta}}\pi(a|s;\boldsymbol{\theta})=0.$$

Therefore, the following equation, which is the same as the definition of the value function $b_{\iota=0}^*(s)$ with the average reward $c_{\iota=0}^*$ as the solution of the Poisson equation [23], can be derived from (14):
$$b_{\iota=0}^*(s)+c_{\iota=0}^*=\mathbb{E}_{\boldsymbol{\theta}}[r+b_{\iota=0}^*(s_{+1})\mid s],\qquad\forall s.\qquad\square$$

# 4 A Generalized NAC Algorithm

We now propose a useful instrumental variable for the gNG($\iota$) estimation and then derive a gNAC algorithm along with an algorithm for $\nabla_{\boldsymbol{\theta}}\ln d_{\boldsymbol{\theta}}(s)$ estimation.

## 4.1 Bias from Estimation of $\nabla_{\boldsymbol{\theta}}\ln d_{\boldsymbol{\theta}}(s)$

For computation of the M-estimator of $\boldsymbol{g}_{\iota,\boldsymbol{\theta}}(s,a;\boldsymbol{\omega})$ as the gNG($\iota$) estimate on $\mathrm{M}(\boldsymbol{\theta})$, the computations of both of the derivatives, $\nabla_{\boldsymbol{\theta}}\ln\pi(a|s;\boldsymbol{\theta})$ and $\nabla_{\boldsymbol{\theta}}\ln d_{\boldsymbol{\theta}}(s)$, are required. While we can easily compute $\nabla_{\boldsymbol{\theta}}\ln\pi(a|s;\boldsymbol{\theta})$ since we have parameterized the policy, we cannot compute the Logarithm stationary State distribution Derivative (LSD) $\nabla_{\boldsymbol{\theta}}\ln d_{\boldsymbol{\theta}}(s)$ analytically unless the state transition probabilities and the reward function are known. Thus, we use the LSD estimate from the algorithm, $\mathcal{LS}$LSD [13]. These LSD estimates $\widehat{\nabla}_{\boldsymbol{\theta}}\ln d_{\boldsymbol{\theta}}(s)$ usually have some estimation errors with finite samples, while the estimates are unbiased, so that $\widehat{\nabla}_{\boldsymbol{\theta}}\ln d_{\boldsymbol{\theta}}(s)=\nabla_{\boldsymbol{\theta}}\ln d_{\boldsymbol{\theta}}(s)+\epsilon(s)$, where $\epsilon(s)$ is an $d$-dimensional random variable satisfying $\mathbb{E}\{\epsilon(s)|s\}=\boldsymbol{0}$.

In such cases, the estimate of gNG($\iota$) from the estimating function (9) or (10) would be biased, because the first condition of (6) for $\boldsymbol{g}_{\iota,\boldsymbol{\theta}}$ is violated unless $\mathbb{E}_{\boldsymbol{\theta}}[\epsilon(s)\epsilon(s)^{\top}]=\boldsymbol{0}$. Thus, in Section 4.2, we consider a refinement of the instrumental variable as the part $\nabla_{\boldsymbol{\theta}}\ln\{d_{\boldsymbol{\theta}}(s)\pi(a|s;\boldsymbol{\theta})\}$ in the estimating function (10), since the instrumental variable can be replaced with any function $\boldsymbol{I}$ that satisfies their conditions[5] for any $s$, $\boldsymbol{\theta}$, and $\boldsymbol{\omega}$ [22] and makes the solution $\boldsymbol{\omega}^*$ become the gNG($\iota$):

$$\mathbb{E}_{\boldsymbol{\theta}}\left[\boldsymbol{I}(r-\{\widehat{\nabla}_{\boldsymbol{\theta}}\ln d_{\boldsymbol{\theta}}(s)^{\iota}+\nabla_{\boldsymbol{\theta}}\ln\pi(a|s;\boldsymbol{\theta})\}^{\top}\boldsymbol{\omega}^*-\rho(s,s_{+1})\right]=\boldsymbol{0},\qquad(16)$$

$$\det|\mathbb{E}_{\boldsymbol{\theta}}[\boldsymbol{I}\{\widehat{\nabla}_{\boldsymbol{\theta}}\ln d_{\boldsymbol{\theta}}(s)^{\iota}+\nabla_{\boldsymbol{\theta}}\ln\pi(a|s;\boldsymbol{\theta})\}^{\top}]|\neq0,\qquad(17)$$

$$\mathbb{E}_{\boldsymbol{\theta}}\left[(r-\{\widehat{\nabla}_{\boldsymbol{\theta}}\ln d_{\boldsymbol{\theta}}(s)^{\iota}+\nabla_{\boldsymbol{\theta}}\ln\pi(a|s;\boldsymbol{\theta})\}^{\top}\boldsymbol{\omega}-\rho(s,s_{+1}))^2\boldsymbol{I}^{\top}\boldsymbol{I}\right]<\infty.\qquad(18)$$

## 4.2 Instrumental variables of near optimal estimating function for gNG($\iota$)

We use a linear function to introduce the auxiliary function (defined in (11)),
$$\rho(s,s_{+1};\boldsymbol{\nu})\triangleq(\tilde{\boldsymbol{\phi}}(s)-[\boldsymbol{\phi}(s_{+1})^{\top}0]^{\top})^{\top}\boldsymbol{\nu},$$

**A Generalized Natural Actor-Critic Algorithm with $\mathcal{LS}$LSD($\lambda$)**

**Given:** A policy $\pi(a|s;\boldsymbol{\theta})$ with an adjustable $\boldsymbol{\theta}$ and a feature vector function of the state, $\boldsymbol{\phi}(s)$.
**Initialize:** $\boldsymbol{\theta}$, $\beta \in [0,1]$, $\alpha$, $\lambda \in [0,1)$,.
**Set:** $\boldsymbol{A} := \boldsymbol{0}$; $\boldsymbol{B} := \boldsymbol{0}$; $\boldsymbol{C} := \boldsymbol{0}$; $\boldsymbol{D} := \boldsymbol{0}$; $\boldsymbol{E} := \boldsymbol{0}$; $\boldsymbol{x} := \boldsymbol{0}$; $\boldsymbol{y} := \boldsymbol{0}$.
**For** $t = 0$ **to** $T-1$ **do**
  **Critic:** Compute the gNG($\iota$) estimate $\widehat{\boldsymbol{\omega}}_\iota$
    $\boldsymbol{A} := \beta\boldsymbol{A} + \nabla_{\boldsymbol{\theta}}\ln\pi(a_t|s_t;\boldsymbol{\theta})\nabla_{\boldsymbol{\theta}}\ln\pi(a_t|s_t;\boldsymbol{\theta})^\top$;   $\boldsymbol{B} := \beta\boldsymbol{B} + \nabla_{\boldsymbol{\theta}}\ln\pi(a_t|s_t;\boldsymbol{\theta})\tilde{\psi}(s_t,s_{t+1})^\top$;
    $\boldsymbol{C} := \beta\boldsymbol{C} + \tilde{\boldsymbol{\phi}}(s_t)\nabla_{\boldsymbol{\theta}}\ln\pi(a_t|s_t;\boldsymbol{\theta})^\top$;   $\boldsymbol{D} := \beta\boldsymbol{D} + \tilde{\boldsymbol{\phi}}(s_t)\boldsymbol{\phi}(s_t)^\top$;   $\boldsymbol{E} := \beta\boldsymbol{E} + \tilde{\boldsymbol{\phi}}(s_t)\tilde{\psi}(s_t,s_{t+1})^\top$;
    $\boldsymbol{x} := \beta\boldsymbol{x} + r_t\nabla_{\boldsymbol{\theta}}\ln\pi(a_t|s_t;\boldsymbol{\theta})$;   $\boldsymbol{y} := \beta\boldsymbol{y} + r_t\tilde{\boldsymbol{\phi}}(s_t)$;   $\boldsymbol{\Omega} :=$ "$\mathcal{LS}$LSD($\lambda$) algorithm" [13]
    $\widehat{\boldsymbol{\omega}}_\iota := \{\boldsymbol{A} + \iota\tilde{\boldsymbol{C}}^\top\boldsymbol{\Omega} - \boldsymbol{B}\boldsymbol{E}^{-1}(\boldsymbol{C} + \iota\boldsymbol{D}\boldsymbol{\Omega})\}^{-1}(\boldsymbol{x} - \boldsymbol{B}\boldsymbol{E}^{-1}\boldsymbol{y})$
  **Actor:** Update $\boldsymbol{\theta}$ by the gNG($\iota$)estimate $\widehat{\boldsymbol{\omega}}$
    $\boldsymbol{\theta} := \boldsymbol{\theta} + \alpha\widehat{\boldsymbol{\omega}}_\iota$;
**End**
**Return:** the policy $\pi(a|s;\boldsymbol{\theta})$.

---

$\ast$ $\tilde{\boldsymbol{C}}$ is the sub-matrix of $\boldsymbol{C}$ getting off the lowest row.

where $\boldsymbol{\nu} \in \mathcal{R}^{|\mathcal{S}|+1}$ and $\boldsymbol{\phi}(s) \in \mathcal{R}^{|\mathcal{S}|}$ are the model parameter and the regressor (feature vector function) of the state $s$, respectively, and $\tilde{\boldsymbol{\phi}}(s) \triangleq [\boldsymbol{\phi}(s)^\top, 1]^\top$. We assume that the set of $\boldsymbol{\phi}(s)$ is linearly independent. Accordingly, the whole model parameter of the estimating function is now $[\boldsymbol{\omega}^\top, \boldsymbol{\nu}^\top]^\top \triangleq \boldsymbol{\varpi}$.

We propose the following instrumental variable

$$\boldsymbol{I}^\star(s,a) \triangleq [\nabla_{\boldsymbol{\theta}}\ln\pi(a|s;\boldsymbol{\theta}), \tilde{\boldsymbol{\phi}}(s)]^\top. \tag{19}$$

Because this instrumental variable $\boldsymbol{I}^\star$ has the desirable property as shown in Theorem 1, the estimating function $\boldsymbol{g}^\star_{\iota,\boldsymbol{\theta}}(s,a;\boldsymbol{\varpi})$ with $\boldsymbol{I}^\star$ is a useful function, even if the LSD is estimated.

**Theorem 1** *To estimate gNG($\iota$), let $\boldsymbol{I}^\star(s,a)$ be used for the estimating function as*

$$\boldsymbol{g}^\star_{\iota,\boldsymbol{\theta}}(s,a;\boldsymbol{\varpi}) = \boldsymbol{I}^\star(s,a)\big\{r - (\widehat{\nabla}_{\boldsymbol{\theta}}\ln d_{\boldsymbol{\theta}}(s)^\iota + \nabla_{\boldsymbol{\theta}}\ln\pi(a|s;\boldsymbol{\theta}))^\top\boldsymbol{\omega} - \rho(s,s_{+1};\boldsymbol{\nu})\big\}, \tag{20}$$

*and $\boldsymbol{\omega}^*$ and $\boldsymbol{\nu}^*$ be the solutions, so $\boldsymbol{\omega}^*$ is equal to the gNG($\iota$), $\boldsymbol{\omega}^* = \widetilde{\nabla}_{\bar{\boldsymbol{F}}_{s,a}(\boldsymbol{\theta},\iota)}\eta(\boldsymbol{\theta})$, and the auxiliary function with $\boldsymbol{\nu}^*$ is the near optimal auxiliary function provided in Lemma 2, $\rho(s,s_{+1};\boldsymbol{\nu}^*) = \rho^*(s,s_{+1})$, even if the LSD estimates include (zero mean) random noises.*

**Proof Sketch:** (i) The condition (18) for the instrumental variable is satisfied due to Assumption 1. (ii) Considering $\mathbb{E}_{\boldsymbol{\theta}}[\widehat{\nabla}_{\boldsymbol{\theta}}\ln d_{\boldsymbol{\theta}}(s)\nabla_{\boldsymbol{\theta}}\ln\pi(a|s;\boldsymbol{\theta})^\top] = \boldsymbol{0}$ and Assumption 1, the condition (17) $\det|\mathbb{E}_{\boldsymbol{\theta}}[\nabla_{\boldsymbol{\varpi}}\boldsymbol{g}^\star_{\iota,\boldsymbol{\theta}}]| \neq \boldsymbol{0}$, is satisfied. This guarantees that the solution $\boldsymbol{\varpi}^* \triangleq [\boldsymbol{\omega}^{*\top}, \boldsymbol{\nu}^{*\top}]^\top$ of (20) that satisfies $\mathbb{E}_{\boldsymbol{\theta}}[\boldsymbol{g}^\star_{\iota,\boldsymbol{\theta}}] = \boldsymbol{0}$ is unique. (iii) Assuming that Theorem 1 is true so that "$\boldsymbol{\omega}^* = \widetilde{\nabla}_{\bar{\boldsymbol{F}}_{s,a}(\boldsymbol{\theta},\iota)}\eta(\boldsymbol{\theta})$" and "$\rho(s,s_{+1};\boldsymbol{\nu}^*) = \rho^*(s,s_{+1})$" hold, then $\mathbb{E}[\boldsymbol{g}^\star_{\iota,\boldsymbol{\theta}}(s,a;\boldsymbol{\varpi}^\top)|s,a]$ becomes $\boldsymbol{I}^\star(s,a)\{r - \overline{R}(s,a)\}$ from (14) and its expectation over $\mathrm{M}(\boldsymbol{\theta})$ becomes equal to $\boldsymbol{0}$. This means that (20) also satisfies the condition (16). From (i), (ii), and (iii), this theorem is proven. $\qquad\square$

The optimal instrumental variable $\boldsymbol{I}^*(s,a)$ with respect to the variance minimization is derived straightforwardly with the results of [21, 24]. However, since $\boldsymbol{I}^*$ is usually to be estimated, we do not adress $\boldsymbol{I}^*$ here. Note that the proposed $\boldsymbol{I}^\star(s,a)$ of (19) can be computed analytically.

### 4.3 A Generalized Natural Actor-Critic Algorithm with $\mathcal{LS}$LSD

We can straightforwardly derive a generalized Natural Actor-Critic algorithm, gNAC($\iota$), by solving the estimating function $\boldsymbol{g}^\star_{\iota,\boldsymbol{\theta}}(s,a;\boldsymbol{\varpi})$ in (20), using the LSD estimate $\widehat{\nabla}_{\boldsymbol{\theta}}\ln d_{\boldsymbol{\theta}}(s) \triangleq \boldsymbol{\Omega}^\top\boldsymbol{\phi}(s)$. However, since $\boldsymbol{\nu}$ in the mode parameter is not required in updating the policy parameter $\boldsymbol{\theta}$, to reduce the computational cost, we compute only $\boldsymbol{\omega}$ by using the results of the block matrices, The above algorithm table shows an instance of the gNAC($\iota$) algorithm with $\mathcal{LS}$LSD($\lambda$) [13] with the forgetting parameter $\beta$ for the statistics, the learning rate of the policy $\alpha$, and the the definitions $\boldsymbol{\psi}(s_{t-1}, s_t) \triangleq [\boldsymbol{\phi}(s_{t-1}) - \boldsymbol{\phi}(s_t)]$ and $\tilde{\boldsymbol{\psi}}(s_{t-1}, s_t) \triangleq [\boldsymbol{\psi}(s_{t-1}, s_t)^\top, 1]^\top$.

Note that the LSD estimate is not used at all in the proposed gNAC($\iota = 0$). In addition, note that gNAC($\iota = 0$) is equivalent to a non-episodic NAC algorithm modified to optimize the average reward, instead of the discounted cumulative reward [1]. This interpretation is consistent with the results of Corollary 1.

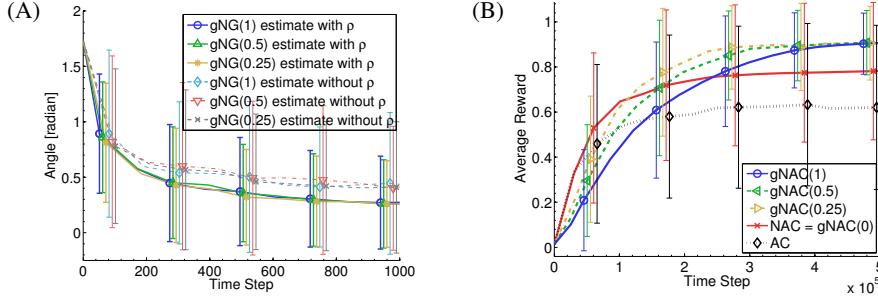

Figure 1: Averages and standard deviations over 50 independent episodes: (A) The angles between the true gNG($\iota$) and estimates with and without the auxiliary function $\rho(s, s_{+1}, \boldsymbol{\nu})$ on the 5-states MDP, (B) The learning performances (average rewards) for the various (N)PGRL algorithms with the auxiliary functions on the 30-states MDP.

## 5    Numerical Experiment

We studied the results of the proposed gNAC algorithm with the various $\iota = \{0, 0.25, 0.5, 1\}$ and randomly synthesized MDPs with $|\mathcal{S}| = \{5, 30\}$ states and $|\mathcal{A}| = 2$ actions. Starting with the performance baseline of the existing PG methods, we used Konda's actor-critic algorithm [23]. This algorithm uses the baseline function in which the state value estimates are estimated by LSTD(0) [25], while the original version did not use any baseline function. Note that gNAC($\iota = 0$) can be regarded as the NAC proposed by [1], which serves as the baseline for the current state-of-the-art PGRL algorithm. We initialized the setting of the MDP in each episode so the set of the actions was always $|\mathcal{A}| = \{l, m\}$. The state transition probability function was set by using the Dirichlet distribution $\mathtt{Dir}(\boldsymbol{\alpha} \in \mathcal{R}^2)$ and the uniform distribution $\mathtt{U}(a; b)$ generating an integer from 1 to $a$ other than $b$: we first initialized it such that $p(s'|s, a) := 0, \forall (s', s, a)$ and then with $\boldsymbol{q}(s, a) \sim \mathtt{Dir}(\boldsymbol{\alpha} = [.3, .3])$ and $s_{\backslash b} \sim \mathtt{U}(|\mathcal{S}|; b)$,

$$\begin{cases} p(s+1|s, l) := q_1(s, l) \\ p(x_{\backslash s+1}|s, l) := q_2(s, l) \end{cases}, \qquad \begin{cases} p(s|s, m) := q_1(s, m) \\ p(x_{\backslash s}|s, m) := q_2(s, m) \end{cases},$$

where $s' = 1$ and $s' = |\mathcal{S}| + 1$ are the identical states. The the reward function $R(s, a, s_{+1})$ was set temporarily with the Gaussian distribution $\mathrm{N}(\mu = 0, \sigma^2 = 1)$, normalized so that $\max_{\boldsymbol{\theta}} \eta(\boldsymbol{\theta}) = 1$ and $\min_{\boldsymbol{\theta}} \eta(\boldsymbol{\theta}) = -1$; $R(s, a, s_{+1}) := 2(R(s, a, s_{+1}) - \min_{\boldsymbol{\theta}} \eta(\boldsymbol{\theta}))/(\max_{\boldsymbol{\theta}} \eta(\boldsymbol{\theta}) - \min_{\boldsymbol{\theta}} \eta(\boldsymbol{\theta})) - 1$. The policy is represented by the sigmoidal function: $\pi(l|s; \boldsymbol{\theta}) = 1/(1 + \exp(-\boldsymbol{\theta}^\top \boldsymbol{\phi}(s)))$. Each $i$th element of the initial policy parameter $\boldsymbol{\theta}_0 \in \mathcal{R}^{|\mathcal{S}|}$ and the feature vector of the $j$th state, $\boldsymbol{\phi}(s_j) \in \mathcal{R}^{|\mathcal{S}|}$, were drawn from $\mathrm{N}(0, 1)$ and $\mathrm{N}(\delta_{\mathrm{ij}}, 0.5)$, respectively, where $\delta_{ij}$ is the Kronecker delta. Figure 1 (A) shows the angles between the true gNG($\iota$) and the gNG($\iota$) estimates with and without the auxiliary function $\rho(s, s_{+1}, \boldsymbol{\nu})$ at $\alpha := 0$ (fixed policy), $\beta := 1$, $\lambda := 0$. The estimation without the auxiliary function was implemented by solving the estimating function of (9). We can confirm that the estimate using $\boldsymbol{g}_{\iota, \boldsymbol{\theta}}^{\star}(s, a; \boldsymbol{\varpi})$ in (20) that implements the near-optimal estimating function became a much more efficient estimator than without the auxiliary function. Figure 1 (B) shows the comparison results in terms of the learning performances, where the learning rates for the gNACs and Konda's actor-critic were set as $\alpha := 3 \times 10^{-4}$ and $\alpha_{\mathrm{Konda}} := 60\alpha$. The other hyper parameters $\beta := 1 - \alpha$ and $\lambda := 0$ were the same in each of the algorithms. We thus confirmed that our gNAC($\iota > 0$) algorithm outperformed the current state-of-the-art NAC algorithm (gNAC($\iota = 0$)).

## 6    Summary

In this paper, we proposed a generalized NG (gNG) learning algorithm that combines two Fisher information matrices for RL. The theory of the estimating function provided insight to prove some important theoretical results from which our proposed gNAC algorithm was derived. Numerical experiments showed that the gNAC algorithm can estimate gNGs efficiently and that it can outperform a current state-of-the-art NAC algorithm. In order to utilize the auxiliary function of the estimating function for the gNG, we defined an auxiliary function on the criterion of the near optimality of the estimating function, by minimizing the distance between the immediate reward as the regressand and the subspace of the regressors of the gNG at the solution of the gNG. However, it may be possible to use different criterion, such as the optimality on the Fisher information matrix metric instead of the Euclidean metric. Also, an analysis of the properties of gNG itself will be necessary to more deeply understand the properties and efficacy of our proposed gNAC algorithm.

## Footnotes

[1]Although to be precise it should be $\Pr(S_{+1} = s_{+1}|S_t = s, A = a)$ for the random variables $S_{+1}$, $S$, and $A$, we write $\Pr(s_{+1}|s, a)$ for simplicity. The same simplification is applied to the other distributions.

[2]The reason for using $\boldsymbol{F}(\boldsymbol{\theta})$ as $\boldsymbol{G}(\boldsymbol{\theta})$ is because the FIM $\boldsymbol{F}_x(\boldsymbol{\theta})$ is a unique metric matrix of the second-order Taylor expansion of the Kullback-Leibler divergence $\Pr(x|\boldsymbol{\theta}+\Delta\boldsymbol{\theta})$ from $\Pr(x|\boldsymbol{\theta})$ [17].

[3]Although there were numerical experiments involving the NSG in [12], they computed the NSG analytically with the state transition probabilities and the reward function, which is typically unknown in RL.

[4]The '*near*' of the near estimating function comes from the approximations of (12) and (13), which implicitly assume that the sum of the (co)variances, $\mathbb{E}_{\boldsymbol{\theta}}[(r - \overline{R}(s,a))^2 + (\rho(s,a,s_{+1}) - \overline{\rho}(s,a))^2 - 2(r - \overline{R}(s,a))(\rho(s,a,s_{+1}) - \overline{\rho}(s,a))|s,a]$, are not large. This assumption seems to hold in many RL tasks.

[5]These correspond to the conditions for the estimating function, (6), (7), and (8).

# References

[1] J. Peters, S. Vijayakumar, and S. Schaal. Natural actor-critic. In *European Conference on Machine Learning*, 2005.

[2] V. Gullapalli. A stochastic reinforcement learning algorithm for learning real-valued functions. *Neural Networks*, 3(6):671–692, 1990.

[3] R. J. Williams. Simple statistical gradient-following algorithms for connectionist reinforcement learning. *Machine Learning*, 8:229–256, 1992.

[4] J. Baxter and P. Bartlett. Infinite-horizon policy-gradient estimation. *Journal of Artificial Intelligence Research*, 15:319–350, 2001.

[5] R. Tedrake, T.W. T. W. Zhang, and H. S. Seung. Stochastic policy gradient reinforcement learning on a simple 3D biped. In *IEEE International Conference on Intelligent Robots and Systems*, 2004.

[6] J. Peters and S. Schaal. Policy gradient methods for robotics. In *IEEE International Conference on Intelligent Robots and Systems*, 2006.

[7] S. Richter, D. Aberdeen, and J. Yu. Natural actor-critic for road traffic optimisation. In *Advances in Neural Information Processing Systems*. MIT Press, 2007.

[8] S. Kakade. A natural policy gradient. In *Advances in Neural Information Processing Systems*, volume 14. MIT Press, 2002.

[9] S. Amari. Natural gradient works efficiently in learning. *Neural Computation*, 10(2):251–276, 1998.

[10] T. Morimura, E. Uchibe, and K. Doya. Utilizing natural gradient in temporal difference reinforcement learning with eligibility traces. In *International Symposium on Information Geometry and its Applications*, pages 256–263, 2005.

[11] S. Bhatnagar, R. Sutton, M. Ghavamzadeh, and M. Lee. Incremental natural actor-critic algorithms. In *Advances in Neural Information Processing Systems*, pages 105–112. MIT Press, 2008.

[12] T. Morimura, E. Uchibe, J. Yoshimoto, and K. Doya. A new natural policy gradient by stationary distribution metric. In *European Conference on Machine Learning and Principles and Practice of Knowledge Discovery in Databases*, 2008.

[13] T. Morimura, E. Uchibe, J. Yoshimoto, J. Peters, and K. Doya. Derivatives of logarithmic stationary distributions for policy gradient reinforcement learning. *Neural Computation*. (in press).

[14] V. Godambe. *Estimating function*. Oxford Science, 1991.

[15] D. P. Bertsekas. *Dynamic Programming and Optimal Control, Volumes 1 and 2*. Athena Scientific, 1995.

[16] R. S. Sutton and A. G. Barto. *Reinforcement Learning*. MIT Press, 1998.

[17] S. Amari and H. Nagaoka. *Method of Information Geometry*. Oxford University Press, 2000.

[18] M. G. Lagoudakis and R. Parr. Least-squares policy iteration. *Journal of Machine Learning Research*, 4:1107–1149, 2003.

[19] D. Bagnell and J. Schneider. Covariant policy search. In *Proceedings of the International Joint Conference on Artificial Intelligence*, July 2003.

[20] S. Amari and M. Kawanabe. Information geometry of estimating functions in semi-parametric statistical models. *Bernoulli*, 3(1), 1997.

[21] A. C. Singh and R. P. Rao. Optimal instrumental variable estimation for linear models with stochastic regressors using estimating functions. In *Symposium on Estimating Functions*, pages 177–192, 1996.

[22] B. Chandrasekhar and B. K. Kale. Unbiased statistical estimating functions in presence of nuisance parameters. *Journal of Statistical Planning and. Inference*, 9:45–54, 1984.

[23] V. S. Konda and J. N. Tsitsiklis. On actor-critic algorithms. *SIAM Journal on Control and Optimization*, 42(4):1143–1166, 2003.

[24] T. Ueno, M. Kawanabe, T. Mori, S. Maeda, and S. Ishii. A semiparametric statistical approach to model-free policy evaluation. In *International Conference on Machine Learning*, pages 857–864, 2008.

[25] J. A. Boyan. Technical update: Least-squares temporal difference learning. *Machine Learning*, 49(2-3):233–246, 2002.

